# Efficient Computation of Complex Distance Metrics Using Hierarchical Filtering

**Patrice Y. Simard**
AT&T Bell Laboratories
Holmdel, NJ 07733

## Abstract

By their very nature, memory based algorithms such as KNN or Parzen windows require a computationally expensive search of a large database of prototypes. In this paper we optimize the searching process for tangent distance (Simard, LeCun and Denker, 1993) to improve speed performance. The closest prototypes are found by recursively searching included subsets of the database using distances of increasing complexity. This is done by using a hierarchy of tangent distances (increasing the number of tangent vectors from 0 to its maximum) and multiresolution (using wavelets). At each stage, a confidence level of the classification is computed. If the confidence is high enough, the computation of more complex distances is avoided. The resulting algorithm applied to character recognition is close to three orders of magnitude faster than computing the full tangent distance on every prototypes.

## 1 INTRODUCTION

Memory based algorithms such as KNN or Parzen windows have been extensively used in pattern recognition. (See (Dasarathy, 1991) for a survey.) Unfortunately, these algorithms often rely on simple distances (such as Euclidean distance, Hamming distance, etc.). As a result, they suffer from high sensitivity to simple transformations of the input patterns that should leave the classification unchanged (e.g. translation or scaling for 2D images). To make the problem worse, these algorithms

are further limited by extensive computational requirements due to the large number of distance computations. (If no optimization technique is used, the computational cost is given in equation 1.)

$$\text{computational cost} \approx \frac{\text{number of}}{\text{prototypes}} \times \frac{\text{distance}}{\text{complexity}} \tag{1}$$

Recently, the problem of transformation sensitivity has been addressed by the introduction of a locally transformation-invariant metric, the tangent distance (Simard, LeCun and Denker, 1993). The basic idea is that instead of measuring the distance $d(A, B)$ between two patterns $A$ and $B$, their respective sets of transformations $T_A$ and $T_B$ are approximated to the first order, and the distance between these two approximated sets is computed. Unfortunately, the tangent distance becomes computationally more expensive as more transformations are taken into consideration, which results in even stronger speed requirements.

The good news is that memory based algorithms are well suited for optimization using hierarchies of prototypes, and that this is even more true when the distance complexity is high. In this paper, we applied these ideas to tangent distance in two ways: 1) Finding the closest prototype can be done by recursively searching included subsets of the database using distances of increasing complexity. This is done by using a hierarchy of tangent distances (increasing the number of tangent vectors from 0 to its maximum) and multiresolution (using wavelets). 2) A confidence level can be computed for each distance. If the confidence in the classification is above a threshold early on, there is no need to compute the more expensive distances. The two methods are described in the next section. Their application on a real world problem will be shown in the result section.

## 2   FILTERING USING A HIERARCHY OF DISTANCES

Our goal is to compute the distance from one unknown pattern to every prototype in a large database in order to determine which one is the closest. It is fairly obvious that some patterns are so different from each other that a very crude approximation of our distance can tell us so. There is a wide range of variation in computation time (and performance) depending on the choice of the distance. For instance, computing the Euclidean distance on $n$-pixel images is a factor $n/k$ of the computation of computing it on $k$-pixels images.

Similarly, at a given resolution, computing the tangent distance with $m$ tangent vectors is $(m + 1)^2$ times as expensive as computing the Euclidean distance ($m = 0$ tangent vectors).

This observations provided us with a hierarchy of about a dozen different distances ranging in computation time from 4 multiply/adds (Euclidean distance on a $2 \times 2$ averaged image) to 20,000 multiply/adds (tangent distance, 7 tangent vectors, $16 \times 16$ pixel images). The resulting filtering algorithm is very straightforward and is exemplified in Figure 1.

The general idea is to store the database of prototypes several times at different resolutions and with different tangent vectors. Each of these resolutions and groups of tangent vectors defines a distance $d_i$. These distances are ordered in increasing

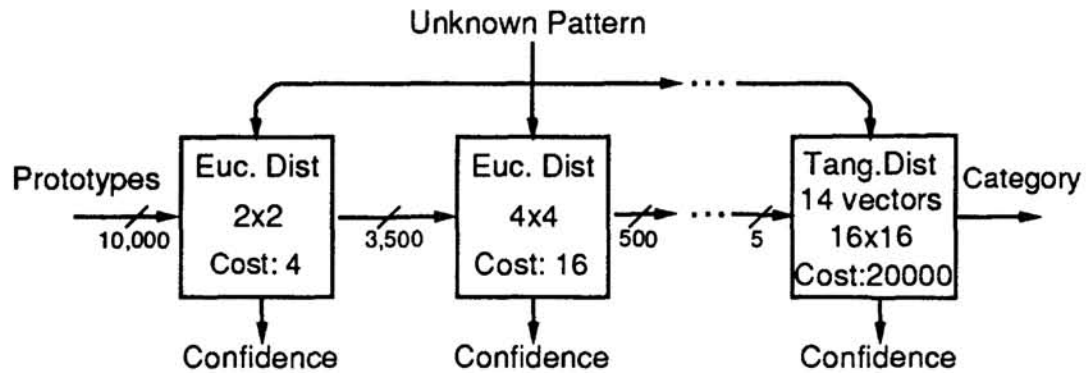

Figure 1: Pattern recognition using a hierarchy of distance. The filter proceed from left (starting with the whole database) to right (where only a few prototypes remain). At each stage distances between prototypes and the unknown pattern are computed, sorted and the best candidate prototypes are selected for the next stage. As the complexity of the distance increases, the number of prototypes decreases, making computation feasible. At each stage a classification is attempted and a confidence score is computed. If the confidence score is high enough, the remaining stages are skipped.

accuracy and complexity. The first distance $d_1$ is computed on all $(K_0)$ prototypes of the database. The closest $K_1$ patterns are then selected and identified to the next stage. This process is repeated for each of the distances; i.e. at each stage $i$, the distance $d_i$ is computed on each $K_{i-1}$ patterns selected by the previous stage. Of course, the idea is that as the complexity of the distance increases, the number of patterns on which this distance must be computed decreases. At the last stage, the most complex and accurate distance is computed on all remaining patterns to determine the classification.

The only difficult part is to determine the minimum $K_i$ patterns selected at each stage for which the filtering does not decrease the overall performance. Note that if the last distance used is the most accurate distance, setting all $K_i$ to the number of patterns in the database will give optimal performance (at the most expensive cost). Increasing $K_i$ always improves the performance in the sense that it allows to find patterns that are closer for the next distance measure $d_{i+1}$. The simplest way to determine $K_i$ is by selecting a validation set and plotting the performance on this validation set as a function of $K_i$. The optimal $K_i$ is then determined graphically. An automatic way of computing each $K_i$ is currently being developed.

This method is very useful when the performance is not degraded by choosing small $K_i$. In this case, the distance evaluation is done using distance metrics which are relatively inexpensive to compute. The computation cost becomes:

$$\text{computational cost} \approx \sum_i \begin{array}{c}\text{number of}\\ \text{prototypes}\\ \text{at stage } i\end{array} \times \begin{array}{c}\text{distance}\\ \text{complexity}\\ \text{at stage } i\end{array} \qquad (2)$$

Curves showing the performance as a function of the value of $K_i$ will be shown in the result section.

# 3    PRUNING THE SEARCH USING CONFIDENCE SCORES

If a confidence score is computed at each stage of the distance evaluation, it is possible for certain patterns to avoid completely computing the most expensive distances. In the extreme case, if the Euclidean distance between two patterns is 0, there is really no need to compute the tangent distance. A simple (and crude) way to compute a confidence score at a given stage $i$, is to find the closest prototype (for distance $d_i$) in each of the possible classes. The distance difference between the closest class and the next closest class gives an approximation of a confidence of this classification. A simple algorithm is then to compare at stage $i$ the confidence score $c_{ip}$ of the current unknown pattern $p$ to a threshold $\theta_i$, and to stop the classification process for this pattern as soon as $c_{ip} > \theta_i$. The classification will then be determined by the closest prototype at this stage. The computation time will therefore be different depending on the pattern to be classified. Easy patterns will be recognized very quickly while difficult patterns will need to be compared to some of the prototypes using the most complex distance. The total computation cost is therefore:

$$\text{computational cost} \approx \sum_i \begin{array}{c}\text{number of}\\ \text{prototypes}\\ \text{at stage } i\end{array} \times \begin{array}{c}\text{distance}\\ \text{complexity}\\ \text{at stage } i\end{array} \times \begin{array}{c}\text{probability}\\ \text{to reach}\\ \text{stage } i\end{array} \qquad (3)$$

Note that if all $\theta_i$ are high, the performance is maximized but so is the cost. We therefore wish to find the smallest value of $\theta_i$ which does not degrade the performance (increasing $\theta_i$ always improves the performance). As in the previous section, the simplest way to determine the optimal $\theta_i$ is graphically with a validation set. Example of curves representing the performance as a function of $\theta_i$ will be given in the result section.

# 4    CHOSING A GOOD HIERARCHY, OPTIMIZATION

## 4.1    k-d tree

Several hierarchies of distance are possible for optimizing the search process. An incremental nearest neighbor search algorithm based on k-d tree (Broder, 1990) was implemented. The k-d tree structure was interesting because it can potentially be used with tangent distance. Indeed, since the separating hyperplanes have n-1 dimension, they can be made parallel to many tangent vectors at the same time. As much as 36 images of 256 pixels with each 7 tangent vectors can be separated into two group of 18 images by one hyperplane which is parallel to all tangent

vectors. The searching algorithm is taking some knowledge of the transformation invariance into account when it computes on which side of each hyperplane the unknown pattern is. Of course, when a leaf is reached, the full tangent distance must be computed.

The problem with the k-d tree algorithm however is that in high dimensional space, the distance from a point to a hyperplane is almost always smaller than the distance between any pair of points. As a result, the unknown pattern must be compared to many prototypes to have a reasonable accuracy. The speed up factor was comparable to our multiresolution approach in the case of Euclidean distance (about 10), but we have not been able to obtain both good performance and high speedup with the k-d tree algorithm applied to tangent distance. This algorithm was not used in our final experiments.

## 4.2    Wavelets

One of the main advantages of the multiresolution approach is that it is easily implemented with wavelet transforms (Mallat, 1989), and that in the wavelet space, the tangent distance is conserved (with orthonormal wavelet bases). Furthermore, the multiresolution decomposition is completely orthogonal to the tangent distance decomposition. In our experiments, the Haar transform was used.

## 4.3    Hierarchy of tangent distance

Many increasingly accurate approximations can be made for the tangent distance at a given resolution. For instance, the tangent distance can be computed by an iterative process of alternative projections onto the tangent hyperplanes. A hierarchy of distances results, derived from the number of projections performed. This hierarchy is not very good because the initial projection is already fairly expensive. It is more desirable to have a better efficiency in the first stages since only few patterns will be left for the latter stages.

Our most successful hierarchy consisted in adding tangent vectors one by one, on both sides. Even though this implies solving a new linear system at each stage, the computational cost is mainly dominated by computing dot products between tangent vectors. These dot-products are then reused in the subsequent stages to create larger linear systems (involving more tangent vectors). This hierarchy has the advantage that the first stage is only twice as expensive, yet much more accurate, than the Euclidean distance. Each subsequent stage brings a lot of accuracy at a reasonable cost. (The cost increases quicker toward the later stages since solving the linear system grows with the cube of the number of tangent vector.) In addition, the last stage is exactly the full tangent distance. As we will see in section 5 the cost in the final stages is negligible.

Obviously, the tangent vectors can be added in different order. We did not try to find the optimal order. For character recognition application adding translations first, followed by hyperbolic deformations, the scalings, the thickness deformations and the rotations yielded good performance.

| $i$ | # of T.V. | Reso | # of proto ($K_i$) | # of prod | Probab | # of mul/add |
|---|---|---|---|---|---|---|
| 0 | 0 | 4 | 9709 | 1 | 1.00 | 40,000 |
| 1 | 0 | 16 | 3500 | 1 | 1.00 | 56,000 |
| 2 | 0 | 64 | 500 | 1 | 1.00 | 32,000 |
| 3 | 1 | 64 | 125 | 2 | 0.90 | 14,000 |
| 4 | 2 | 256 | 50 | 5 | 0.60 | 40,000 |
| 5 | 4 | 256 | 45 | 7 | 0.40 | 32,000 |
| 6 | 6 | 256 | 25 | 9 | 0.20 | 11,000 |
| 7 | 8 | 256 | 15 | 11 | 0.10 | 4,000 |
| 8 | 10 | 256 | 10 | 13 | 0.10 | 3,000 |
| 9 | 12 | 256 | 5 | 15 | 0.05 | 1,000 |
| 10 | 14 | 256 | 5 | 17 | 0.05 | 1,000 |

Table 1: Summary computation for the classification of 1 pattern: The first column is the distance index, the second column indicates the number of tangent vector (0 for the Euclidean distance), and the third column indicates the resolution in pixels, the fourth is $K_i$ or the number of prototypes on which the distance $d_i$ must be computed, the fifth column indicates the number of additional dot products which must be computed to evaluate distance $d_i$, the sixth column indicates the probability to not skip that stage after the confidence score has been used, and the last column indicates the total average number of multiply-adds which must be performed (product of column 3 to 6) at each stage.

## 4.4    Selecting the $k$ closests out of $N$ prototypes in O(N)

In the multiresolution filter, at the early stages we must select the $k$ closest prototypes from a large number of prototypes. This is problematic because the prototypes cannot be sorted since $O(N log N)$ is expensive compared to computing $N$ distances at very low resolution (like 4 pixels). A simple solution consists in using a variation of "quicksort" or "finding the k-th element" (Aho, Hopcroft and Ullman, 1983), which can select the $k$ closests out of $N$ prototypes in O(N). The generic idea is to compute the mean of the distances (an approximation is actually sufficient) and then to split the distances

into two halves (of different sizes) according to whether they are smaller or larger than the mean distance. If they are more distances smaller than the mean than $k$, the process is reiterated on the upper half, otherwise it is reiterated on the lower half. The process is recursively executed until there is only one distance in each half. ($k$ is then reached and all the $k$ prototypes in the lower halves are closer to the unknown pattern than all the $N - k$ prototypes in the upper halves.) Note that the elements are not sorted and that only the expected time is O(N), but this is sufficient for our problem.

## 5    RESULTS

A simple task of pattern classification was used to test the filtering. The prototype set and the test set consisted respectively of 9709 and 2007 labeled images (16 by 16 pixels) of handwritten digits. The prototypes were also averaged to lower

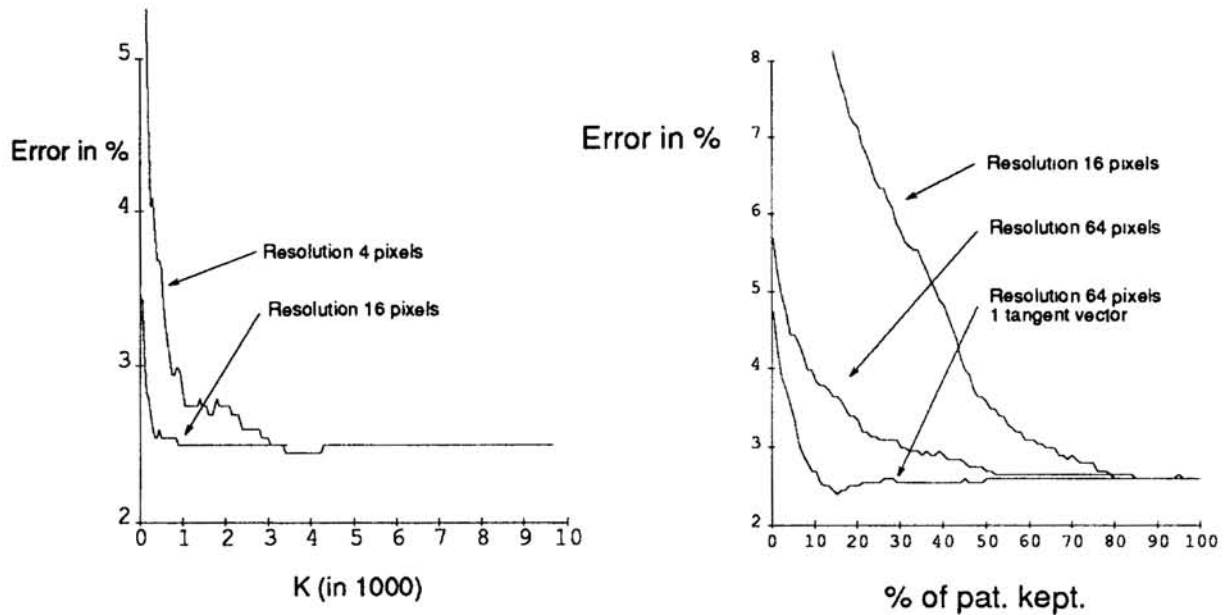

Figure 2: Left: Raw error performance as a function of $K_1$ and $K_2$. The final chosen values were $K_1 = 3500$ and $K_2 = 500$. Right: Raw error as a function of the percentage of pattern which have not exceeded the confidence threshold $\theta_i$. A 100% means all the pattern were passed to the next stage.

resolutions (2 by 2, 4 by 4 and 8 by 8) and copied to separate databases. The 1 by 1 resolution was not useful for anything. Therefore the fastest distance was the Euclidean distance on 2 by 2 images, while the slowest distance was the full tangent distance with 7 tangent vectors for both the prototype and the unknown pattern (Simard, LeCun and Denker, 1993). Table 1 summarizes the results.

Several observations can be made. First, simple distance metrics are very useful to eliminate large proportions of prototypes at no cost in performances. Indeed the Euclidean distance computed on 2 by 2 images can remove 2 third of the prototypes. Figure 2, left, shows the performance as a function of $K_1$ and $K_2$ (2.5 % raw error was considered optimal performance). It can be noticed that for $K_i$ above a certain value, the performance is optimal and constant. The most complex distances (6 and 7 tangent vectors on each side) need only be computed for 5% of the prototypes.

The second observation is that the use of a confidence score can greatly reduce the number of distance evaluations in later stages. For instance the dominant phases of the computation would be with 2, 4 and 6 tangent vectors at resolution 256 if there were not reduced to 60%, 40% and 20% respectively using the confidence scores. Figure 2, right, shows the raw error performance as a function of the percentage of rejection (confidence lower than $\theta_i$) at stage $i$. It can be noticed that above a certain threshold, the performance are optimal and constant. Less than 10% of the unknown patterns need the most complex distances (5, 6 and 7 tangent vectors on each side), to be computed.

# 6   DISCUSSION

Even though our method is by no way optimal (the order of the tangent vector can be changed, intermediate resolution can be used, etc...), the overall speed up we achieved was about 3 orders of magnitude (compared with computing the full tangent distance on all the patterns). There was no significant decrease in performances. This classification speed is comparable with neural network method, but the performance are better with tangent distance (2.5% versus 3%). Furthermore the above methods require no learning period which makes them very attractive for application were the distribution of the patterns to be classified is changing rapidly.

The hierarchical filtering can also be combined with learning the prototypes using algorithms such as learning vector quantization (LVQ).

# References

Aho, A. V., Hopcroft, J. E., and Ullman, J. D. (1983). *Data Structure and Algorithms.* Addison-Wesley.

Broder, A. J. (1990). Strategies for Efficient Incremental Nearest Neighbor Search. *Pattern Recognition*, 23:171–178.

Dasarathy, B. V. (1991). *Nearest Neighbor (NN) Norms: NN Pattern classification Techniques.* IEEE Computer Society Press, Los Alamitos, California.

Mallat, S. G. (1989). A Theory for Multiresolution Signal Decomposition: The Wavelet Representation. *IEEE Transactions on Pattern Analysis and Machine Intelligence*, 11, No. 7:674–693.

Simard, P. Y., LeCun, Y., and Denker, J. (1993). Efficient Pattern Recognition Using a New Transformation Distance. In *Neural Information Processing Systems*, volume 4, pages 50–58, San Mateo, CA.
